# A Parallel Mixture of SVMs for Very Large Scale Problems

**Ronan Collobert**[*]
Université de Montréal, DIRO
CP 6128, Succ. Centre-Ville
Montréal, Québec, Canada
collober@iro.umontreal.ca

**Samy Bengio**
IDIAP
CP 592, rue du Simplon 4
1920 Martigny, Switzerland
bengio@idiap.ch

**Yoshua Bengio**
Université de Montréal, DIRO
CP 6128, Succ. Centre-Ville
Montréal, Québec, Canada
bengioy@iro.umontreal.ca

## Abstract

Support Vector Machines (SVMs) are currently the state-of-the-art models for many classification problems but they suffer from the complexity of their training algorithm which is at least *quadratic* with respect to the number of examples. Hence, it is hopeless to try to solve real-life problems having more than a few hundreds of thousands examples with SVMs. The present paper proposes a new mixture of SVMs that can be easily implemented in parallel and where each SVM is trained on a small subset of the whole dataset. Experiments on a large benchmark dataset (Forest) as well as a difficult speech database, yielded significant time improvement (time complexity appears empirically to locally grow *linearly* with the number of examples). In addition, and that is a surprise, a significant improvement in *generalization* was observed on Forest.

## 1 Introduction

Recently a lot of work has been done around Support Vector Machines [9], mainly due to their impressive generalization performances on classification problems when compared to other algorithms such as artificial neural networks [3, 6]. However, SVMs require to solve a quadratic optimization problem which needs resources that are at least quadratic in the number of training examples, and it is thus hopeless to try solving problems having millions of examples using classical SVMs.

In order to overcome this drawback, we propose in this paper to use a mixture of several SVMs, each of them trained only on a part of the dataset. The idea of an SVM mixture is not new, although previous attempts such as Kwok's paper on Support Vector Mixtures [5] did not train the SVMs on part of the dataset but on the whole dataset and hence could not overcome the

---

[*]Part of this work has been done while Ronan Collobert was at IDIAP, CP 592, rue du Simplon 4, 1920 Martigny, Switzerland.

time complexity problem for large datasets. We propose here a *simple method* to train such a mixture, and we will show that *in practice* this method is *much faster* than training only one SVM, and leads to results that are *at least as good as one SVM*. We conjecture that the training time complexity of the proposed approach with respect to the number of examples is sub-quadratic for large data sets. Moreover this mixture can be easily parallelized, which could improve again *significantly* the training time.

The organization of the paper goes as follows: in the next section, we briefly introduce the SVM model for classification. In section 3 we present our mixture of SVMs, followed in section 4 by some comparisons to related models. In section 5 we show some experimental results, first on a toy dataset, then on two large real-life datasets. A short conclusion then follows.

## 2  Introduction to Support Vector Machines

Support Vector Machines (SVMs) [9] have been applied to many classification problems, generally yielding good performance compared to other algorithms. The decision function is of the form

$$y = \text{sign}\left(\sum_{i=1}^{N} y_i \alpha_i K(\boldsymbol{x}, \boldsymbol{x}_i) + b\right) \tag{1}$$

where $\boldsymbol{x} \in \mathbb{R}^d$ is the d-dimensional input vector of a test example, $y \in \{-1, 1\}$ is a class label, $\boldsymbol{x}_i$ is the input vector for the $i^{th}$ training example, $y_i$ is its associated class label, $N$ is the number of training examples, $K(\boldsymbol{x}, \boldsymbol{x}_i)$ is a positive definite kernel function, and $\boldsymbol{\alpha} = \{\alpha_1, \ldots, \alpha_N\}$ and $b$ are the parameters of the model. Training an SVM consists in finding $\boldsymbol{\alpha}$ that minimizes the objective function

$$Q(\boldsymbol{\alpha}) = -\sum_{i=1}^{N} \alpha_i + \frac{1}{2}\sum_{i=1}^{N}\sum_{j=1}^{N} \alpha_i \alpha_j y_i y_j K(\boldsymbol{x}_i, \boldsymbol{x}_j) \tag{2}$$

subject to the constraints

$$\sum_{i=1}^{N} \alpha_i y_i = 0 \tag{3}$$

and

$$0 \le \alpha_i \le C \quad \forall i. \tag{4}$$

The kernel $K(\boldsymbol{x}, \boldsymbol{x}_i)$ can have different forms, such as the Radial Basis Function (RBF):

$$K(\boldsymbol{x}_i, \boldsymbol{x}_j) = \exp\left(\frac{-\|\boldsymbol{x}_i - \boldsymbol{x}_j\|^2}{\sigma^2}\right) \tag{5}$$

with parameter $\sigma$.

Therefore, to train an SVM, we need to solve a quadratic optimization problem, where the number of parameters is $N$. This makes the use of SVMs for large datasets difficult: computing $K(x_i, x_j)$ for every training pair would require $O(N^2)$ computation, and solving may take up to $O(N^3)$. Note however that current state-of-the-art algorithms appear to have training time complexity scaling much closer to $O(N^2)$ than $O(N^3)$ [2].

## 3  A New Conditional Mixture of SVMs

In this section we introduce a new type of mixture of SVMs. The output of the mixture for an input vector $\boldsymbol{x}$ is computed as follows:

$$f(\boldsymbol{x}) = h\left(\sum_{m=1}^{M} w_m(\boldsymbol{x}) s_m(\boldsymbol{x})\right) \tag{6}$$

where $M$ is the number of experts in the mixture, $s_m(\boldsymbol{x})$ is the output of the $m^{th}$ expert given input $\boldsymbol{x}$, $w_m(\boldsymbol{x})$ is the weight for the $m^{th}$ expert given by a "gater" module taking also $\boldsymbol{x}$ in input, and $h$ is a transfer function which could be for example the hyperbolic tangent for classification tasks. Here each expert is an SVM, and we took a neural network for the gater in our experiments. In the proposed model, the gater is trained to minimize the cost function

$$C = \sum_{i=1}^{N} [f(\boldsymbol{x}_i) - y_i]^2 . \tag{7}$$

To train this model, we propose a very simple algorithm:

1. Divide the training set into $M$ random subsets of size near $N/M$.

2. Train each expert separately over one of these subsets.

3. Keeping the experts fixed, train the gater to minimize (7) on the whole training set.

4. Reconstruct $M$ subsets: for each example $(\boldsymbol{x}_i, y_i)$,
   - sort the experts in descending order according to the values $w_m(\boldsymbol{x}_i)$,
   - assign the example to the first expert in the list which has less than $(N/M + c)$ examples*, in order to ensure a balance between the experts.

5. If a termination criterion is not fulfilled (such as a given number of iterations or a validation error going up), goto step 2.

Note that step 2 of this algorithm can be easily implemented in parallel as each expert can be trained separately on a different computer. Note also that step 3 can be an approximate minimization (as usually done when training neural networks).

## 4   Other Mixtures of SVMs

The idea of mixture models is quite old and has given rise to very popular algorithms, such as the well-known *Mixture of Experts* [4] where the cost function is similar to equation (7) but where the gater and the experts are trained, using gradient descent or EM, on the whole dataset (and not subsets) and their parameters are trained simultaneously. Hence such an algorithm is quite demanding in terms of resources when the dataset is large, if training time scales like $O(N^p)$ with $p > 1$.

In the more recent *Support Vector Mixture* model [5], the author shows how to replace the experts (typically neural networks) by SVMs and gives a learning algorithm for this model. Once again the resulting mixture is trained jointly on the whole dataset, and hence does not solve the quadratic barrier when the dataset is large.

In another *divide-and-conquer* approach [7], the authors propose to first divide the training set using an unsupervised algorithm to cluster the data (typically a mixture of Gaussians), then train an expert (such as an SVM) on each subset of the data corresponding to a cluster, and finally recombine the outputs of the experts. Here, the algorithm does indeed train separately the experts on small datasets, like the present algorithm, but there is no notion of a loop reassigning the examples to experts according to the prediction made by the gater of how well each expert performs on each example. Our experiments suggest that this element is essential to the success of the algorithm.

Finally, the *Bayesian Committee Machine* [8] is a technique to partition the data into several subsets, train SVMs on the individual subsets and then use a specific combination scheme based on the covariance of the test data to combine the predictions. This method scales linearly in the

number of training data, but is in fact a *transductive* method as it cannot operate on a single test example. Like in the previous case, this algorithm assigns the examples randomly to the experts (however the Bayesian framework would in principle allow to find better assignments).

Regarding our proposed mixture of SVMs, if the number of experts grows with the number of examples, and the number of outer loop iterations is a constant, then the total training time of the experts scales linearly with the number of examples. Indeed, given $N$ the total number of examples, choose the number of expert $M$ such that the ratio $\frac{N}{M}$ is a constant $r$; Then, if $k$ is the number of outer loop iterations, and if the training time for an SVM with $r$ examples is $O(r^\beta)$ (empirically $\beta$ is slightly above 2), the total training time of the experts is $O(kr^\beta * M) = O(kr^{\beta-1}N)$, where $k$, $r$ and $\beta$ are constants, which gives a total training time of $O(N)$. In particular for $\beta = 2$ that gives $O(krN)$. The actual total training time should however also include $k$ times the training time of the gater, which may potentially grow more rapidly than $O(N)$. However, it did not appear to be the case in our experiments, thus yielding apparent linear training time. Future work will focus on methods to reduce the gater training time and guarantee linear training time per outer loop iteration.

## 5   Experiments

In this section, we present three sets of experiments comparing the new mixture of SVMs to other machine learning algorithms. Note that all the SVMs in these experiments have been trained using *SVMTorch* [2].

### 5.1   A Toy Problem

In the first series of experiments, we first tested the mixture on an artificial toy problem for which we generated 10,000 training examples and 10,000 test examples. The problem had two non-linearly separable classes and had two input dimensions. On Figure 1 we show the decision surfaces obtained first by a linear SVM, then by a Gaussian SVM, and finally by the proposed mixture of SVMs. Moreover, in the latter, the gater was a simple linear function and there were two linear SVMs in the mixture[†]. This artificial problem thus shows clearly that the algorithm seems to work, and is able to combine, even linearly, very simple models in order to produce a non-linear decision surface.

### 5.2   A Large-Scale Realistic Problem: Forest

For a more realistic problem, we did a series of experiments on part of the *UCI Forest* dataset[‡]. We modified the 7-class classification problem into a binary classification problem where the goal was to separate class 2 from the other 6 classes. Each example was described by 54 input features, each normalized by dividing by the maximum found on the training set. The dataset had more than 500,000 examples and this allowed us to prepare a series of experiments as follows:

- We kept a separate test set of 50,000 examples to compare the best mixture of SVMs to other learning algorithms.
- We used a validation set of 10,000 examples to select the best mixture of SVMs, varying the number of experts and the number of hidden units in the gater.
- We trained our models on different training sets, using from 100,000 to 400,000 examples.
- The mixtures had from 10 to 50 expert SVMs with Gaussian kernel and the gater was an MLP with between 25 and 500 hidden units.

---

[†]Note that the transfer function $h()$ was still a tanh().

[‡]The Forest dataset is available on the UCI website at the following address: ftp://ftp.ics.uci.edu/pub/machine-learning-databases/covtype/covtype.info.

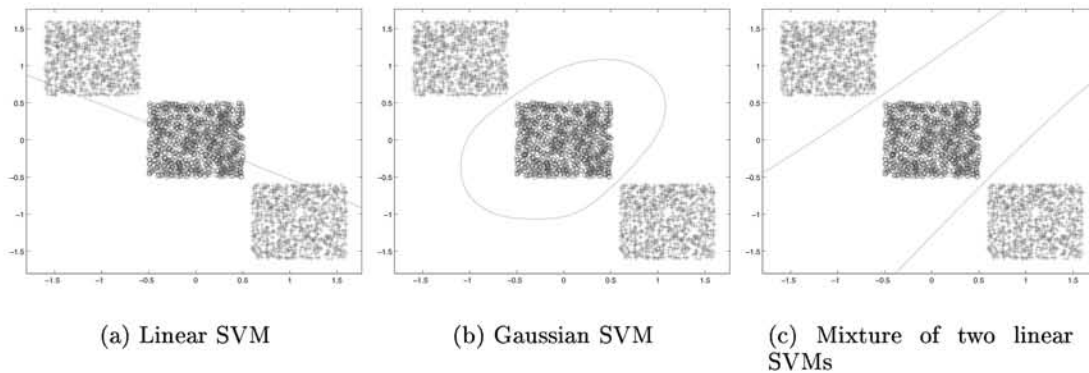

| (a) Linear SVM | (b) Gaussian SVM | (c) Mixture of two linear SVMs |
|---|---|---|

Figure 1: Comparison of the decision surfaces obtained by (a) a linear SVM, (b) a Gaussian SVM, and (c) a linear mixture of two linear SVMs, on a two-dimensional classification toy problem.

Note that since the number of examples was quite large, we selected the internal training parameters such as the $\sigma$ of the Gaussian kernel of the SVMs or the learning rate of the gater using a held-out portion of the training set. We compared our models to

- a single MLP, where the number of hidden units was selected by cross-validation between 25 and 250 units,
- a single SVM, where the parameter of the kernel was also selected by cross-validation,
- a mixture of SVMs where the gater was replaced by a constant vector, assigning the same weight value to every expert.

Table 1 gives the results of a first series of experiments with a fixed training set of 100,000 examples. To select among the variants of the gated SVM mixture we considered performance over the validation set as well as training time. All the SVMs used $\sigma = 1.7$. The selected model had 50 experts and a gater with 150 hidden units. A model with 500 hidden units would have given a performance of 8.1% over the test set but would have taken 621 minutes on one machine (and 388 minutes on 50 machines).

|  | Train Error (%) | Test | Time (minutes) (1 cpu) | (50 cpu) |
|---|---|---|---|---|
| one MLP | 17.56 | 18.15 | 12 |  |
| one SVM | 16.03 | 16.76 | 3231 |  |
| uniform SVM mixture | 19.69 | 20.31 | 85 | 2 |
| gated SVM mixture | 5.91 | 9.28 | 237 | 73 |

Table 1: Comparison of performance between an MLP (100 hidden units), a single SVM, a uniform SVM mixture where the gater always output the same value for each expert, and finally a mixture of SVMs as proposed in this paper.

As it can be seen, the gated SVM outperformed all models in terms of training and test error. Note that the training error of the single SVM is high because its hyper-parameters were selected to minimize error on the validation set (other values could yield to much lower training error but larger test error). It was also much faster, even on one machine, than the SVM and since the mixture could easily be parallelized (each expert can be trained separately), we also reported

the time it took to train on 50 machines. In a first attempt to understand these results, one can at least say that the power of the model does not lie only in the MLP gater, since a single MLP was pretty bad, it is neither only because we used SVMs, since a single SVM was not as good as the gated mixture, and it was not only because we divided the problem into many sub-problems since the uniform mixture also performed badly. It seems to be a combination of all these elements.

We also did a series of experiments in order to see the influence of the number of hidden units of the gater as well as the number of experts in the mixture. Figure 2 shows the validation error of different mixtures of SVMs, where the number of hidden units varied from 25 to 500 and the number of experts varied from 10 to 50. There is a clear performance improvement when the number of hidden units is increased, while the improvement with additional experts exists but is not as strong. Note however that the training time increases also rapidly with the number of hidden units while it slightly decreases with the number of experts if one uses one computer per expert.

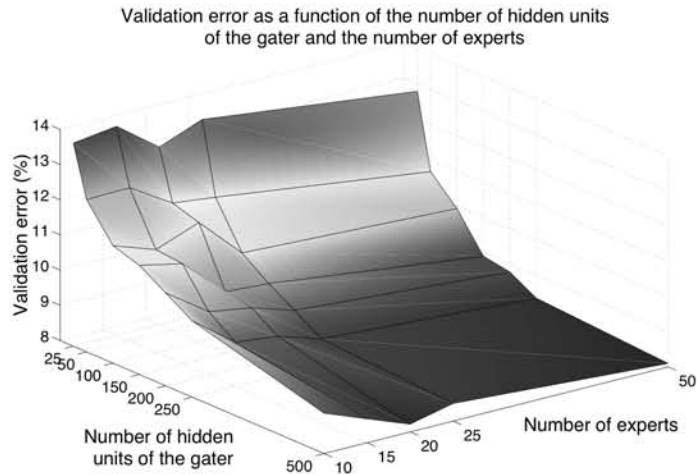

Figure 2: Comparison of the validation error of different mixtures of SVMs with various number of hidden units and experts.

In order to find how the algorithm scaled with respect to the number of examples, we then compared the same mixture of experts (50 experts, 150 hidden units in the gater) on different training set sizes. Table 3 shows the validation error of the mixture of SVMs trained on training sets of sizes from 100,000 to 400,000. It seems that, at least in this range and for this particular dataset, the mixture of SVMs scales linearly with respect to the number of examples, and not quadratically as a classical SVM. It is interesting to see for instance that the mixture of SVMs was able to solve a problem of 400,000 examples in less than 7 hours (on 50 computers) while it would have taken more than one month to solve the same problem with a single SVM.

Finally, figure 4 shows the evolution of the training and validation errors of a mixture of 50 SVMs gated by an MLP with 150 hidden units, during 5 iterations of the algorithm. This should convince that the loop of the algorithm is essential in order to obtain good performance. It is also clear that the empirical convergence of the outer loop is extremely rapid.

## 5.3   Verification on Another Large-Scale Problem

In order to verify that the results obtained on *Forest* were replicable on other large-scale problems, we tested the SVM mixture on a speech task. We used the *Numbers95* dataset [1] and

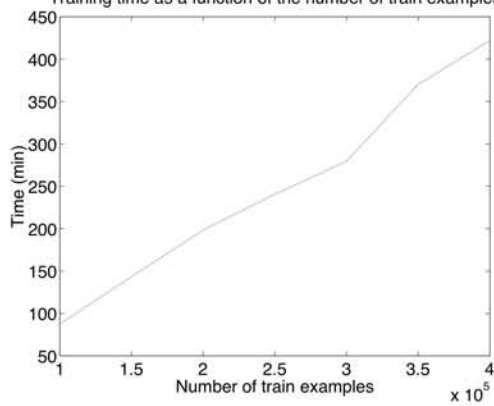

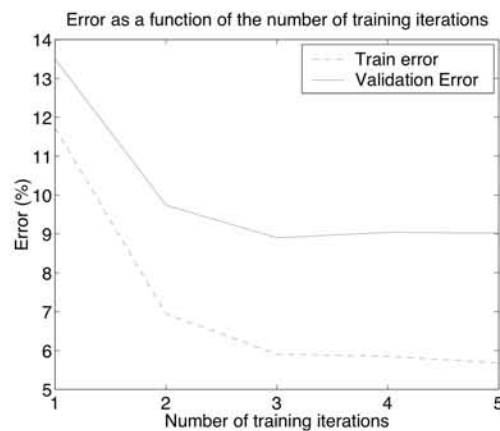

Figure 3: Comparison of the training time of the same mixture of SVMs (50 experts, 150 hidden units in the gater) trained on different training set sizes, from 100,000 to 400,000.

Figure 4: Comparison of the training and validation errors of the mixture of SVMs as a function of the number of training iterations.

turned it into a binary classification problem where the task was to separate silence frames from non-silence frames. The total number of frames was around 540,000 frames. The training set contained 100,000 randomly chosen frames out of the first 400,000 frames. The disjoint validation set contained 10,000 randomly chosen frames out of the first 400,000 frames also. Finally, the test set contained 50,000 randomly chosen frames out of the last 140,000 frames. Note that the validation set was used here to select the number of experts in the mixture, the number of hidden units in the gater, and $\sigma$. Each frame was parameterized using standard methods used in speech recognition (j-rasta coefficients, with first and second temporal derivatives) and was thus described by 45 coefficients, but we used in fact an input window of three frames, yielding 135 input features per examples.

Table 2 shows a comparison between a single SVM and a mixture of SVMs on this dataset. The number of experts in the mixture was set to 50, the number of hidden units of the gater was set to 50, and the $\sigma$ of the SVMs was set to 3.0. As it can be seen, the mixture of SVMs was again many times faster than the single SVM (even on 1 cpu only) but yielded similar generalization performance.

|  | Train Error (%) | Test | Time (minutes) (1 cpu) | (50 cpu) |
|---|---|---|---|---|
| one SVM | 0.98 | 7.57 | 6787 | |
| gated SVM mixture | 4.41 | 7.32 | 851 | 65 |

Table 2: Comparison of performance between a single SVM and a mixture of SVMs on the speech dataset.

## 6 Conclusion

In this paper we have presented a new algorithm to train a mixture of SVMs that gave very good results compared to classical SVMs either in terms of training time or generalization performance on two large scale difficult databases. Moreover, the algorithm appears to scale linearly with the number of examples, at least between 100,000 and 400,000 examples.

These results are extremely encouraging and suggest that the proposed method could allow training SVM-like models for very large multi-million data sets in a reasonable time. If training of the neural network gater with stochastic gradient takes time that grows much less than quadratically, as we conjecture it to be the case for very large data sets (to reach a "good enough" solution), then the whole method is clearly sub-quadratic in training time with respect to the number of training examples. Future work will address several questions: how to guarantee linear training time for the gater as well as for the experts? can better results be obtained by tuning the hyper-parameters of each expert separately? Does the approach work well for other types of experts?

## Acknowledgments

RC would like to thank the Swiss NSF for financial support (project FN2100-061234.00). YB would like to thank the NSERC funding agency and NCM$^2$ network for support.

## Footnotes

*where $c$ is a small positive constant. In the experiments, $c = 1$.

## References

[1] R.A. Cole, M. Noel, T. Lander, and T. Durham. New telephone speech corpora at CSLU. *Proceedings of the European Conference on Speech Communication and Technology, EU-ROSPEECH*, 1:821–824, 1995.

[2] R. Collobert and S. Bengio. SVMTorch: Support vector machines for large-scale regression problems. *Journal of Machine Learning Research*, 1:143–160, 2001.

[3] C. Cortes and V. Vapnik. Support vector networks. *Machine Learning*, 20:273–297, 1995.

[4] Robert A. Jacobs, Michael I. Jordan, Steven J. Nowlan, and Geoffrey E. Hinton. Adaptive mixtures of local experts. *Neural Computation*, 3(1):79–87, 1991.

[5] J. T. Kwok. Support vector mixture for classification and regression problems. In *Proceedings of the International Conference on Pattern Recognition (ICPR)*, pages 255–258, Brisbane, Queensland, Australia, 1998.

[6] E. Osuna, R. Freund, and F. Girosi. Training support vector machines: an application to face detection. In *IEEE conference on Computer Vision and Pattern Recognition*, pages 130–136, San Juan, Puerto Rico, 1997.

[7] A. Rida, A. Labbi, and C. Pellegrini. Local experts combination trough density decomposition. In *International Workshop on AI and Statistics (Uncertainty'99)*. Morgan Kaufmann, 1999.

[8] V. Tresp. A bayesian committee machine. *Neural Computation*, 12:2719–2741, 2000.

[9] V. N. Vapnik. *The nature of statistical learning theory*. Springer, second edition, 1995.
